# Linearly constrained Bayesian matrix factorization for blind source separation

**Mikkel N. Schmidt**
Department of Engineering
University of Cambridge
mns@imm.dtu.dk

## Abstract

We present a general Bayesian approach to probabilistic matrix factorization subject to linear constraints. The approach is based on a Gaussian observation model and Gaussian priors with bilinear equality and inequality constraints. We present an efficient Markov chain Monte Carlo inference procedure based on Gibbs sampling. Special cases of the proposed model are Bayesian formulations of non-negative matrix factorization and factor analysis. The method is evaluated on a blind source separation problem. We demonstrate that our algorithm can be used to extract meaningful and interpretable features that are remarkably different from features extracted using existing related matrix factorization techniques.

## 1 Introduction

Source separation problems arise when a number of signals are mixed together, and the objective is to estimate the underlying sources based on the observed mixture. In the supervised, model-based approach to source separation, examples of isolated sources are used to train source models, which are then combined in order to separate a mixture. Oppositely, in unsupervised, blind source separation, only very general information about the sources is available. Instead of estimating models of the sources, blind source separation is based on relatively weak criteria such as minimizing correlations, maximizing statistical independence, or fitting data subject to constraints.

Under the assumptions of linear mixing and additive noise, blind source separation can be expressed as a matrix factorization problem,

$$\underset{I \times J}{\boldsymbol{X}} = \underset{I \times K}{\boldsymbol{A}} \, \underset{K \times J}{\boldsymbol{B}} + \underset{I \times J}{\boldsymbol{N}}, \quad \text{or equivalently,} \quad x_{ij} = \sum_{k=1}^{K} a_{ik} b_{kj} + n_{ij}, \tag{1}$$

where the subscripts below the matrices denote their dimensions. The columns of $\boldsymbol{A}$ represent $K$ unknown sources, and the elements of $\boldsymbol{B}$ are the mixing coefficients. Each of the $J$ columns of $\boldsymbol{X}$ contains an observation that is a mixture of the sources plus additive noise represented by the columns of $\boldsymbol{N}$. The objective is to estimate the sources, $\boldsymbol{A}$, as well as the mixing coefficients, $\boldsymbol{B}$, when only the data matrix, $\boldsymbol{X}$, is observed. In a Bayesian formulation, the aim is not to compute a single value for $\boldsymbol{A}$ and $\boldsymbol{B}$, but to infer their posterior distribution under a set of model assumptions. These assumptions are specified in the likelihood function, $p(\boldsymbol{X}|\boldsymbol{A}, \boldsymbol{B})$, which expresses the probability of the data given the factorizing matrices, and in the prior, $p(\boldsymbol{A}, \boldsymbol{B})$, which describes available knowledge before observing the data. Depending on the specific choice of likelihood and priors, matrix factorizations with different characteristics can be devised.

Non-negative matrix factorization (NMF), which is distinguished from other matrix factorization techniques by its non-negativity constraints, has been shown to decompose data into meaningful, interpretable parts [3]; however, a parts-based decomposition is not necessarily useful, unless it

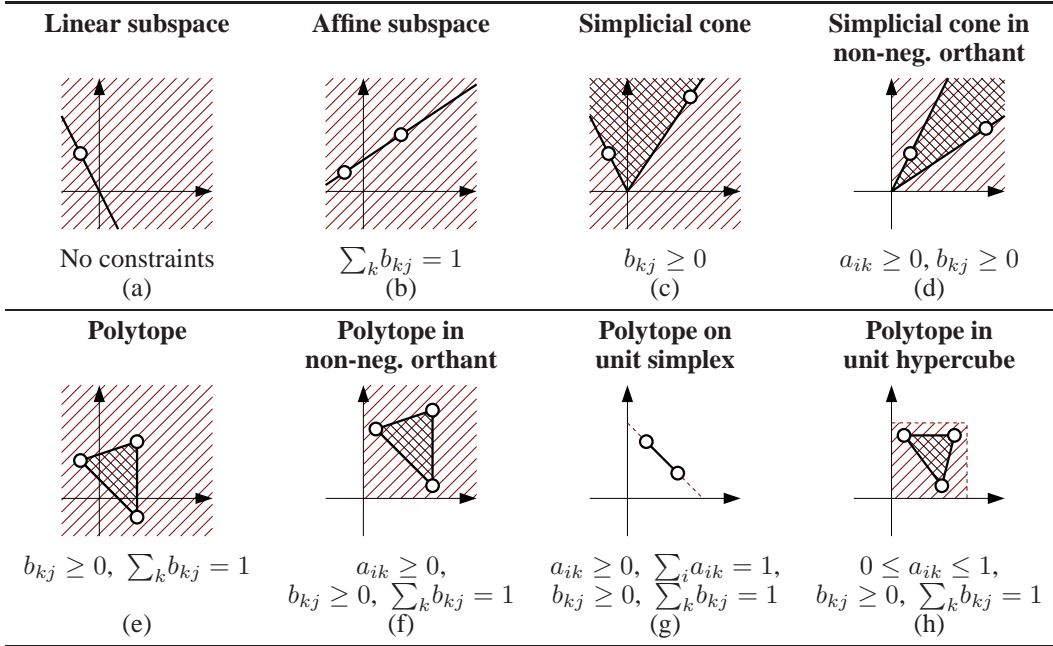

| Linear subspace | Affine subspace | Simplicial cone | Simplicial cone in non-neg. orthant |
|---|---|---|---|
| No constraints | $\sum_k b_{kj} = 1$ | $b_{kj} \geq 0$ | $a_{ik} \geq 0, b_{kj} \geq 0$ |
| (a) | (b) | (c) | (d) |

| Polytope | Polytope in non-neg. orthant | Polytope on unit simplex | Polytope in unit hypercube |
|---|---|---|---|
| $b_{kj} \geq 0, \ \sum_k b_{kj} = 1$ | $a_{ik} \geq 0,$ $b_{kj} \geq 0, \ \sum_k b_{kj} = 1$ | $a_{ik} \geq 0, \ \sum_i a_{ik} = 1,$ $b_{kj} \geq 0, \ \sum_k b_{kj} = 1$ | $0 \leq a_{ik} \leq 1,$ $b_{kj} \geq 0, \ \sum_k b_{kj} = 1$ |
| (e) | (f) | (g) | (h) |

Figure 1: Examples of model spaces that can be attained using matrix factorization with different linear constraints in $A$ and $B$. The red hatched area indicates the feasible region for the source vectors (columns of $A$). Dots, ○, are examples of specific positions of source vectors, and the black hatched area, is the corresponding feasible region for the data vectors. Special cases include (a) factor analysis and (d) non-negative matrix factorization.

finds the "correct" parts. The main contribution in this paper is that specifying relevant constraints other than non-negativity substantially changes the qualities of the results obtained using matrix factorization. Some intuition about how the incorporation of different constraints affects the matrix factorization can be gained by considering their geometric implications. Figure 1 shows how different linear constraints on $A$ and $B$ constrain the model space. For example, if the mixing coefficients are constrained to be non-negative, data is modelled as the convex hull of a simplicial cone, and if the mixing coefficients are further constrained to sum to unity, data is modelled as the hull of a convex polytope.

In this paper, we develop a general and flexible framework for Bayesian matrix factorization, in which the unknown sources and the mixing coefficients are treated as hidden variables. Furthermore, we allow any number of linear equality or inequality constraints to be specified. On an unsupervised image separation problem, we demonstrate, that when relevant constraints are specified, the method finds interpretable features that are remarkably different from features computed using other matrix factorization techniques.

The proposed method is related to recently proposed Bayesian matrix factorization techniques: Bayesian matrix factorization based on Gibbs sampling has been demonstrated [7, 8] to scale up to very large datasets and to avoid the problem of overfitting associated with non-Bayesian techniques. Bayesian methods for non-negative matrix factorization have also been proposed, either based on variational inference [1] or Gibbs sampling [4, 9]. The latter can be seen as special cases of the algorithm proposed here.

The paper is structured as follows: In section 2, the linearly constrained Bayesian matrix factorization model is described. Section 3 presents an inference procedure based on Gibbs sampling. In Section 4, the method is applied to an unsupervised source separation problem and compared to other existing matrix factorization methods. We discuss our results and conclude in Section 5.

## 2 The linearly constrained Bayesian matrix factorization model

In the following, we describe the linearly constrained Bayesian matrix factorization model. We make specific choices for the likelihood and priors that keep the formulation general while allowing for efficient inference based on Gibbs sampling.

### 2.1 Noise model

We choose an iid. zero mean Gaussian noise model,

$$p(n_{ij}) = \mathcal{N}(n_{ij}|0, v_{ij}) = \frac{1}{\sqrt{2\pi v_{ij}}} \exp\left(-\frac{n_{ij}^2}{2v_{ij}}\right), \tag{2}$$

where, in the most general formulation, each matrix element has its own variance, $v_{ij}$; however, the variance parameters can easily be joined, e.g., to have a single noise variance per row or just one overall variance, which corresponds to an isotropic noise model. The noise model gives rise to the likelihood, i.e., the probability of the observations given the parameters of the model. The likelihood is given by

$$p(\boldsymbol{x}|\boldsymbol{\theta}) = \prod_{i=1}^{I}\prod_{j=1}^{J} \mathcal{N}\left(x_{ij} \Big| \sum_{k=1}^{K} a_{ik}b_{kj}, v_{ij}\right) = \prod_{i=1}^{I}\prod_{j=1}^{J} \frac{1}{\sqrt{2\pi v_{ij}}} \exp\left(-\frac{(x - \sum_{k=1}^{K} a_{ik}b_{kj})^2}{2v_{ij}}\right), \tag{3}$$

where $\boldsymbol{\theta} = \{\boldsymbol{A}, \boldsymbol{B}, \{v_{ij}\}\}$ denotes all parameters in the model. For the noise variance parameters we choose conjugate inverse-gamma priors,

$$p(v_{ij}) = \mathcal{IG}(v_{ij}|\alpha, \beta) = \frac{\beta^{\alpha}}{\Gamma(\alpha)} v_{ij}^{-(\alpha+1)} \exp\left(\frac{-\beta}{v_{ij}}\right). \tag{4}$$

### 2.2 Priors for sources and mixing coefficients

We now define the prior distribution for the factorizing matrices, $\boldsymbol{A}$ and $\boldsymbol{B}$. To simplify the notation, we specify the matrices by vectors $\boldsymbol{a} = \text{vec}(\boldsymbol{A}^{\top}) = [a_{11}, a_{12}, \ldots, a_{IK}]^{\top}$ and $\boldsymbol{b} = \text{vec}(\boldsymbol{B}) = [b_{11}, b_{21}, \ldots, b_{KJ}]^{\top}$. We choose a Gaussian prior over $\boldsymbol{a}$ and $\boldsymbol{b}$ subject to inequality constraints, $\mathcal{Q}$, and equality constraints, $\mathcal{R}$,

$$p(\boldsymbol{a}, \boldsymbol{b}) \propto \begin{cases} \mathcal{N}\left(\begin{bmatrix} \boldsymbol{a} \\ \boldsymbol{b} \end{bmatrix} \Big| \underbrace{\begin{bmatrix} \boldsymbol{\mu}_a \\ \boldsymbol{\mu}_b \end{bmatrix}}_{\equiv \boldsymbol{\mu}}, \underbrace{\begin{bmatrix} \boldsymbol{\Sigma}_a & \boldsymbol{\Sigma}_{ab} \\ \boldsymbol{\Sigma}_{ab}^{\top} & \boldsymbol{\Sigma}_b \end{bmatrix}}_{\equiv \boldsymbol{\Sigma}}\right), & \text{if } \mathcal{Q}(\boldsymbol{a}, \boldsymbol{b}) \leq \boldsymbol{0}, \ \mathcal{R}(\boldsymbol{a}, \boldsymbol{b}) = \boldsymbol{0}, \\ 0, & \text{otherwise.} \end{cases} \tag{5}$$

In slight abuse of denotation, we refer to $\boldsymbol{\mu}$ and $\boldsymbol{\Sigma}$ as the mean and covariance matrix, although the actual mean and covariance of $\boldsymbol{a}$ and $\boldsymbol{b}$ depends on the constraints.

In the most general formulation, the constraints, $\mathcal{Q}: \mathbb{R}^{IK} \times \mathbb{R}^{KJ} \rightarrow \mathbb{R}^{N_{\mathcal{Q}}}$ and $\mathcal{R}: \mathbb{R}^{IK} \times \mathbb{R}^{KJ} \rightarrow \mathbb{R}^{N_{\mathcal{R}}}$, are biaffine maps, that define $N_{\mathcal{Q}}$ inequality and $N_{\mathcal{R}}$ equality constraints jointly in $\boldsymbol{a}$ and $\boldsymbol{b}$. Specifically, each inequality constraint has the form

$$\mathcal{Q}_m(\boldsymbol{a}, \boldsymbol{b}) = q_m + \boldsymbol{a}^{\top}\boldsymbol{q}_m^{(a)} + \boldsymbol{b}^{\top}\boldsymbol{q}_m^{(b)} + \boldsymbol{a}^{\top}\boldsymbol{Q}_m^{(ab)}\boldsymbol{b} \leq 0. \tag{6}$$

By rearranging terms and combining the $N_{\mathcal{Q}}$ constraints in matrix notation, we may write

$$\underbrace{\left[\boldsymbol{q}_1^{(a)} + \boldsymbol{Q}_1^{(ab)}\boldsymbol{b} \ \cdots \ \boldsymbol{q}_{N_{\mathcal{Q}}}^{(a)} + \boldsymbol{Q}_{N_{\mathcal{Q}}}^{(ab)}\boldsymbol{b}\right]^{\top}}_{\equiv \boldsymbol{Q}_a} \boldsymbol{a} \leq \underbrace{\begin{bmatrix} -q_1 - \boldsymbol{b}^{\top}\boldsymbol{q}_1^{(b)} \\ \vdots \\ -q_{N_{\mathcal{Q}}} - \boldsymbol{b}^{\top}\boldsymbol{q}_{N_{\mathcal{Q}}}^{(b)} \end{bmatrix}}_{\equiv \boldsymbol{q}_a}, \qquad \boldsymbol{Q}_a^{\top}\boldsymbol{a} \leq \boldsymbol{q}_a, \tag{7}$$

from which it is clear that the constraints are linear in $\boldsymbol{a}$. Likewise, the constraints can be rearranged to a linear form in $\boldsymbol{b}$. The equality constraints, $\mathcal{R}$, are defined analogously.

This general formulation of the priors allows all elements of $\boldsymbol{a}$ and $\boldsymbol{b}$ to have prior dependencies both through their covariance matrix, $\boldsymbol{\Sigma}_{ab}$, and through the joint constraints; however, in some

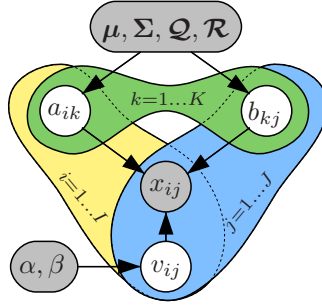

Figure 2: Graphical model for linearly constrained Bayesian matrix factorization, when $A$ and $B$ are independent in the prior. White and grey nodes represent latent and observed variables respectively, and arrows indicate stochastic dependensies. The colored plates denote repeated variables over the indicated indices.

applications it is not relevant or practical to specify all of these dependencies in advance. We may restrict the model such that $a$ and $b$ are independent a priori by setting $\Sigma_{ab}$, $Q_m^{(ab)}$, and $R_m^{(ab)}$ to zero, and restricting $q_m^{(a)} = 0$ for all $m$ where $q_m^{(b)} \neq 0$ and vice versa. Furthermore, we can decouple the elements of $A$, or groups of elements such as rows or columns, by choosing $\Sigma_a$, $Q_a$, and $R_a$ to have an appropriate block structure. Similarly we can decouple elements of $B$.

## 2.3  Posterior distribution

Having specified the model and the prior densities, we can now write the posterior, which is the distribution of the parameters conditioned on the observed data and hyper-parameters. The posterior is given by

$$p(\boldsymbol{\theta}|\boldsymbol{x},\boldsymbol{\psi}) \propto p(\boldsymbol{a},\boldsymbol{b})p(\boldsymbol{x}|\boldsymbol{\theta}) \prod_{i=1}^{I}\prod_{j=1}^{J} p(v_{ij}), \tag{8}$$

where $\boldsymbol{\psi} = \{\alpha, \beta, \boldsymbol{\mu}, \boldsymbol{\Sigma}, \boldsymbol{Q}, \boldsymbol{R}\}$ denotes all hyper-parameters in the model. A graphical representation of the model is given in Figure 2.

## 3  Inference

In a Bayesian framework, we are interested in computing the posterior distribution over the parameters, $p(\boldsymbol{\theta}|\boldsymbol{x},\boldsymbol{\psi})$. The posterior, given in Eq. (8), is only known up to a multiplicative constant, and direct computation of this normalizing constant involves integrating over the unnormalized posterior, which is not analytically tractable. Instead, we approximate the posterior distribution using Markov chain Monte Carlo (MCMC).

### 3.1  Gibbs sampling

We propose an inference procedure based on Gibbs sampling. Gibbs sampling is applicable when the joint density of the parameters is not known, but the parameters can be partitioned into groups, such that their posterior conditional densities are known. We iteratively sweep through the groups of parameters and generate a random sample for each, conditioned on the current value of the others. This procedure forms a homogenous Markov chain and its stationary distribution is exactly the joint posterior.

In the following, we derive the posterior conditional densities required in the Gibbs sampler. First, we consider the noise variances, $v_{ij}$. Due to the choice of conjugate prior, the posterior density is an inverse-gamma,

$$p(v_{ij}|\boldsymbol{\theta}\backslash v_{ij}) = \mathcal{IG}(v_{ij}|\bar{\alpha},\bar{\beta}), \tag{9}$$

$$\bar{\alpha} = \alpha + \tfrac{1}{2}, \quad \bar{\beta} = \beta + \tfrac{1}{2}\big(x_{ij} - \textstyle\sum_{k=1}^{K} a_{ik}b_{kj}\big)^2, \tag{10}$$

from which samples can be generated using standard acceptance-rejection methods.

Next, we consider the factorizing matrices, represented by the vectors $\boldsymbol{a}$ and $\boldsymbol{b}$. We only discuss generating samples from $\boldsymbol{a}$, since the sampling procedure for $\boldsymbol{b}$ is identical due to the symmetry of the model. Conditioned on $\boldsymbol{b}$, the prior density of $\boldsymbol{a}$ is a constrained Gaussian,

$$p(\boldsymbol{a}|\boldsymbol{b}) \propto \begin{cases} \mathcal{N}(\boldsymbol{a}|\tilde{\boldsymbol{\mu}}_a, \tilde{\boldsymbol{\Sigma}}_a), & \text{if } \boldsymbol{Q}_a^\top \boldsymbol{a} \leq \boldsymbol{q}_a, \ \boldsymbol{R}_a^\top \boldsymbol{a} = \boldsymbol{r}_a, \\ 0, & \text{otherwise,} \end{cases} \tag{11}$$

$$\tilde{\boldsymbol{\mu}}_a = \boldsymbol{\mu}_a + \boldsymbol{\Sigma}_{ab}\boldsymbol{\Sigma}_b^{-1}(\boldsymbol{b} - \boldsymbol{\mu}_b), \quad \tilde{\boldsymbol{\Sigma}}_a = \boldsymbol{\Sigma}_a - \boldsymbol{\Sigma}_{ab}\boldsymbol{\Sigma}_b^{-1}\boldsymbol{\Sigma}_{ab}^\top, \tag{12}$$

where we have used Eq. (7) and the standard result for a conditional Gaussian density. In the special case when $\boldsymbol{a}$ and $\boldsymbol{b}$ are independent in the prior, we simply have $\tilde{\boldsymbol{\mu}}_a = \boldsymbol{\mu}_a$ and $\tilde{\boldsymbol{\Sigma}}_a = \boldsymbol{\Sigma}_a$. Further, conditioning on the data leads to the final expression for the posterior conditional density of $\boldsymbol{a}$,

$$p(\boldsymbol{a}|\boldsymbol{x}, \boldsymbol{\theta}\backslash\boldsymbol{a}) \propto \begin{cases} \mathcal{N}(\boldsymbol{a}|\bar{\boldsymbol{\mu}}_a, \bar{\boldsymbol{\Sigma}}_a), & \text{if } \boldsymbol{Q}_a^\top \boldsymbol{a} \leq \boldsymbol{q}_a, \ \boldsymbol{R}_a^\top \boldsymbol{a} = \boldsymbol{r}_a, \\ 0, & \text{otherwise,} \end{cases} \tag{13}$$

$$\bar{\boldsymbol{\Sigma}}_a = \left(\tilde{\boldsymbol{\Sigma}}_a^{-1} + \tilde{\boldsymbol{B}}\boldsymbol{V}^{-1}\tilde{\boldsymbol{B}}^\top\right)^{-1}, \quad \bar{\boldsymbol{\mu}}_a = \bar{\boldsymbol{\Sigma}}_a\left(\tilde{\boldsymbol{\Sigma}}_a^{-1}\tilde{\boldsymbol{\mu}}_a + \tilde{\boldsymbol{B}}\boldsymbol{V}^{-1}\boldsymbol{x}\right), \tag{14}$$

where $\boldsymbol{V} = \mathrm{diag}(v_{11}, v_{12}, \ldots, v_{IJ})$ and $\tilde{\boldsymbol{B}} = \mathrm{diag}(\boldsymbol{B}, \ldots, \boldsymbol{B})$ is a diagonal block matrix with $I$ repetitions of $\boldsymbol{B}$.

The Gibbs sampler proceeds iteratively: First, the noise variance is generated from the inverse-gamma density in Eq. (9); second, $\boldsymbol{a}$ is generated from the constrained Gaussian density in Eq. (13); and third, $\boldsymbol{b}$ is generated from a constrained Gaussian analogous to Eq. (13).

## 3.2 Sampling from a constrained Gaussian

An essential component in the proposed matrix factorization method is an algorithm for generating random samples from a multivariate Gaussian density subject to linear equality and inequality constraints. With a slight change of notation, we consider generating $\boldsymbol{x} \in \mathbb{R}^N$ from the density

$$p(\boldsymbol{x}) \propto \begin{cases} \mathcal{N}(\boldsymbol{x}|\boldsymbol{\mu}_x, \boldsymbol{\Sigma}_x), & \text{if } \boldsymbol{Q}_x^\top \boldsymbol{x} \leq \boldsymbol{q}_x, \ \boldsymbol{R}_x^\top \boldsymbol{x} = \boldsymbol{r}_x, \\ 0, & \text{otherwise.} \end{cases} \tag{15}$$

A similar problem has previously been treated by Geweke [2], who proposes a Gibbs sampling procedure, that does not handle equality constraints and no more than $N$ inequality constraints. Rodriguez-Yam et al. [6] extends the method in [2] to an arbitrary number of inequality constraints, but do not provide an algorithm for handling equality constraints. Here, we present a general Gibbs sampling procedure that handles any number of equality and inequality constraints.

The equality constraints restrict the distribution to an affine subspace of dimensionality $N - R$, where $R$ is the number of linearly independent constraints. The conditional distribution on that subspace is a Gaussian subject to inequality constraints. To handle the equality constraints, we map the distribution onto this subspace. Using the singular value decomposition (SVD), we can robustly compute an orthonormal basis, $\boldsymbol{T}$, for the constraints, as well as its orthogonal complement, $\boldsymbol{T}_\perp$,

$$\boldsymbol{R}_x = \boldsymbol{U}\boldsymbol{S}\boldsymbol{V}^\top = \begin{bmatrix} \boldsymbol{T} \\ \boldsymbol{T}_\perp \end{bmatrix}^\top \begin{bmatrix} \boldsymbol{S}_T & \boldsymbol{0} \\ \boldsymbol{0} & \boldsymbol{0} \end{bmatrix} \boldsymbol{V}^\top, \tag{16}$$

where $\boldsymbol{S}_T = \mathrm{diag}(s_1, \ldots, s_R)$ holds the $R$ non-zero singular values. We now define a transformed variable, $\boldsymbol{y}$, that is related to $\boldsymbol{x}$ by

$$\boldsymbol{y} = \boldsymbol{T}_\perp(\boldsymbol{x} - \boldsymbol{x}_0), \quad \boldsymbol{y} \in \mathbb{R}^{N-R} \tag{17}$$

where $\boldsymbol{x}_0$ is some vector that satisfies the equality constraints, e.g., computed using the pseudo-inverse, $\boldsymbol{x}_0 = \boldsymbol{R}_x^{\dagger\top}\boldsymbol{r}_x$. This transformation ensures, that for any value of $\boldsymbol{y}$, the corresponding $\boldsymbol{x}$ satisfies the equality constraints. We can now compute the distribution of $\boldsymbol{y}$ conditioned on the equality constraints, which is Gaussian subject to inequality constraints,

$$p(\boldsymbol{y}|\boldsymbol{R}_x^\top \boldsymbol{x} = \boldsymbol{r}_x) \propto \begin{cases} \mathcal{N}(\boldsymbol{y}|\boldsymbol{\mu}_y, \boldsymbol{\Sigma}_y) & \text{if } \boldsymbol{Q}_y^\top \boldsymbol{y} \leq \boldsymbol{q}_y \\ 0 & \text{otherwise,} \end{cases} \tag{18}$$

$$\boldsymbol{\mu}_y = \boldsymbol{\Lambda}(\boldsymbol{\mu}_x - \boldsymbol{x}_0), \quad \boldsymbol{\Sigma}_y = \boldsymbol{\Lambda}\boldsymbol{\Sigma}_x\boldsymbol{T}_\perp^\top, \quad \boldsymbol{Q}_y = \boldsymbol{T}_\perp\boldsymbol{Q}_x, \quad \boldsymbol{q}_y = \boldsymbol{q}_x - \boldsymbol{Q}_x^\top\boldsymbol{x}_0, \tag{19}$$

where $\mathbf{\Lambda} = \boldsymbol{T}_\perp(\boldsymbol{I} - \boldsymbol{\Sigma}_x\boldsymbol{T}^\top(\boldsymbol{T}\boldsymbol{\Sigma}_x\boldsymbol{T}^\top)^{-1}\boldsymbol{T})$.

We introduce a second transformation with the purpose of reducing the correlations between the variables. This may potentially improve the sampling procedure, because Gibbs sampling can suffer from slow mixing when the distribution is highly correlated. Correlations between the elements of $\boldsymbol{y}$ are due to both the Gaussian covariance structure and the inequality constraints; however, for simplicity we only decorrelate with respect to the covariance of the underlying unconstrained Gaussian. To this end, we define the transformed variable, $\boldsymbol{z}$, given by

$$\boldsymbol{z} = \boldsymbol{L}^{-\top}(\boldsymbol{y} - \boldsymbol{\mu}_y), \tag{20}$$

where $\boldsymbol{L}$ is the Cholesky factorization of the covariance matrix, $\boldsymbol{L}\boldsymbol{L}^\top = \boldsymbol{\Sigma}_y$. The distribution of $\boldsymbol{z}$ is then a standard Gaussian subject to inequality constraints,

$$p(\boldsymbol{z}|\boldsymbol{R}_x^\top\boldsymbol{x} = \boldsymbol{r}_x) \propto \begin{cases} \mathcal{N}(\boldsymbol{z}|\boldsymbol{0}, \boldsymbol{I}), & \text{if } \boldsymbol{Q}_z^\top\boldsymbol{z} \leq \boldsymbol{q}_z, \\ 0, & \text{otherwise}, \end{cases} \tag{21}$$

$$\boldsymbol{Q}_z = \boldsymbol{L}\boldsymbol{Q}_y, \quad \boldsymbol{q}_z = \boldsymbol{q}_y - \boldsymbol{Q}_y^\top\boldsymbol{\mu}_y. \tag{22}$$

We can now sample from $\boldsymbol{z}$ using a Gibbs sampling procedure by sweeping over the elements $z_i$ and generating samples from their conditional distributions, which are univariate truncated standard Gaussian,

$$p(z_i|\boldsymbol{z}\backslash z_i) = \frac{\sqrt{\frac{2}{\pi}}\exp\left(\frac{-z_i^2}{2}\right)}{\operatorname{erf}\left(\frac{u_i}{\sqrt{2}}\right) - \operatorname{erf}\left(\frac{\ell_i}{\sqrt{2}}\right)} \propto \begin{cases} \mathcal{N}(z_i|0, 1), & \ell_i \leq z_i \leq u_i, \\ 0, & \text{otherwise}. \end{cases} \tag{23}$$

Samples from this density can be generated using standard methods such as inverse transform sampling (transforming a uniform random variable by the inverse cumulative density function); the efficient mixed rejection sampling algorithm proposed by Geweke [2]; or slice sampling [5].

The upper and lower points of truncation can be computed as

$$\boldsymbol{Q}_z^\top\boldsymbol{z} \quad \leq \quad \boldsymbol{q}_z \tag{24}$$

$$\underbrace{[\boldsymbol{Q}_z]_{i:}^\top z_i}_{d} \quad \leq \quad \underbrace{\boldsymbol{q}_z - [\boldsymbol{Q}_z]_{\bar{i}:}^\top\boldsymbol{z}_{\bar{i}}}_{n} \tag{25}$$

$$\ell_i = \max\left\{-\infty, \tfrac{n_k}{d_k} : d_k < 0\right\} \leq z_i \leq \min\left\{\infty, \tfrac{n_k}{d_k} : d_k > 0\right\} = u_i, \tag{26}$$

where $[\boldsymbol{Q}_z]_{i:}$ denotes the $i$th row of $\boldsymbol{Q}_z$, $[\boldsymbol{Q}_z]_{\bar{i}:}$ denotes all rows except the $i$th, and $\boldsymbol{z}_{\bar{i}}$ denotes the vector of all elements of $\boldsymbol{z}$ except the $i$th.

Finally, when a sample of $\boldsymbol{z}$ has been generated after a number of Gibbs sweeps, it can be transformed into a sample of the original variable, $\boldsymbol{x}$, using

$$\boldsymbol{x} = \boldsymbol{T}_\perp^\top(\boldsymbol{L}^\top\boldsymbol{z} + \boldsymbol{\mu}_y) + \boldsymbol{x}_0. \tag{27}$$

The sampling procedure is illustrated in Figure 3.

## 4 Experiments

We demonstrate the proposed linearly constrained Bayesian matrix factorization method on a blind image separation problem, and compare it to two other matrix factorization techniques: independent component analysis (ICA) and non-negative matrix factorization (NMF).

**Data**  We used a subset from the MNIST dataset which consists of $28 \times 28$ pixel grayscale images of handwritten digits (see Figure 4.a). We selected the first 800 images of each digit, 0–9, which gave us $8,000$ unique images. From these images we created $4,000$ image mixtures by adding the grayscale intensities of the images two and two, such that the different digits were combined in equal proportion. We rescaled the mixed images so that their pixel intensities were in the 0–1 interval, and arranged the vectorized images as the columns of the matrix $\boldsymbol{X} \in \mathbb{R}^{I \times J}$, where $I = 784$ and $J = 4,000$. Examples of the image mixtures are shown in Figure 4.b.

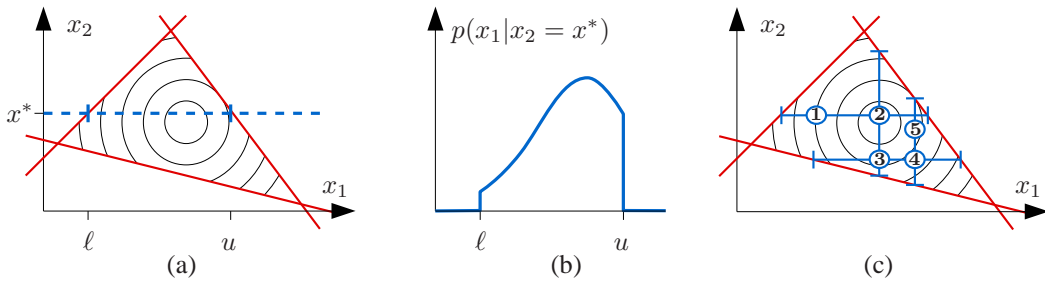

Figure 3: Gibbs sampling from a multivariate Gaussian density subject to linear constraints. a) Two-dimensional Gaussian subject to three inequality constraints. b) The conditional distribution of $x_1$ given $x_2 = x^*$ is a truncated Gaussian. c) Gibbs sampling proceeds iteratively by sweeping over the dimensions and sampling from the conditional distribution in each dimension conditioned on the current value in the other dimensions.

**Task**   The objective is to factorize the data matrix in order to find a number of source images that explain the data. Ideally, the sources should correspond to the original digits. We cannot hope to find exactly 10 sources that each corresponds to a digit, because there are large variations as to how each digit is written. For that reason, we used 40 hidden sources in our experiments, which allowed 4 exemplars on average for each digit.

**Method**   For comparison we factorized the mixed image data using two standard matrix factorization techniques: ICA, where we used the FastICA algorithm, and NMF, where we used Lee and Seung's multiplicative update algorithm [3]. The sources determined using these methods are shown in Figure 4.c–d.

For the linearly constrained Bayesian matrix factorization, we used an isotropic noise model. We chose a decoupled prior for $A$ and $B$ with zero mean, $\mu = 0$, and unit diagonal covariance matrix, $\Sigma = I$. The hidden sources were constrained to have the same range of pixel intensities as the image mixtures, $0 \leq a_{ik} \leq 1$. This constraint allows the sources to be interpreted as images since pixel intensities outside this interval are not meaningful. The mixing coefficients were constrained to be non-negative, $b_{kj} \geq 0$, and sum to unity, $\sum_{k=1}^{K} b_{kj} = 1$; thus, the observed data is modeled as a convex combination of the sources. The constraints ensure that only additive combinations of the sources are allowed, and introduces a negative correlation between the mixing coefficients. This negative correlation implies that if one source contributes more to a mixture the other sources must correspondingly contribute less. The idea behind this constraint is that it will lead the sources to *compete* as opposed to *collaborate* to explain the data. A geometric interpretation of the constraints is illustrated in Figure 1.h: The data vectors are modeled by a convex polytope in the non-negative unit hypercube, and the hidden sources are the vertices of this polytope. We computed $10,000$ Gibbs samples, which appeared sufficient for the sampler to converge. The result of the matrix factorization are shown in Figure 4.e, which displays a single sample of $A$ at the last iteration.

**Results**   In ICA (see Figure 4.c) the sources are not constrained to be non-negative, and therefore do not have a direct interpretation as images. Most of the computed sources are complex patterns, that appear to be dominated by a single digit. While ICA certainly does find structure in the data, the estimated sources lack a clear interpretation.

The sources computed using NMF (see Figure 4.d) have the property which Lee and Seung [3] refer to as a *parts-based representation*. Spatially, the sources are *local* as opposed to *global*. The decomposition has an intuitive interpretation: Each source is a short line segment or a dot, and the different digits can be constructed by combining these parts.

Linearly constrained Bayesian matrix factorization (see Figure 4.e) computes sources with a very clear and intuitive interpretation: Almost all of the $40$ computed sources visually resemble handwritten digits, and are thus well aligned with the sources that were used to generate the mixtures. Compared to the original data, the computed sources are a bit bolder and have slightly smeared

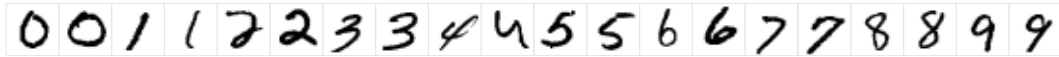

(a) Original dataset: MNIST digits

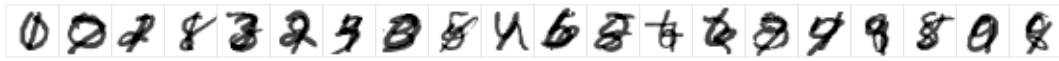

(b) Training data: Mixture of digits

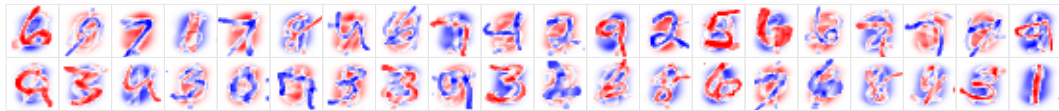

(c) Independent component analysis

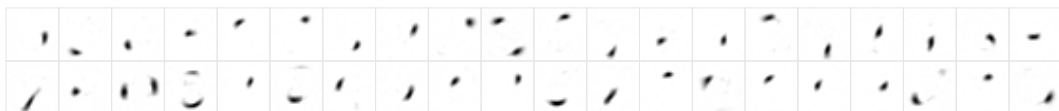

(d) Non-negative matrix factorization

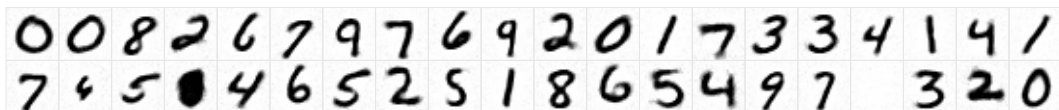

(e) Linearly constrained Bayesian matrix factorization

Figure 4: Data and results of the analyses of an image separation problem. a) The MNIST digits data (20 examples shown) used to generate the mixture data. b) The mixture data consists of 4000 images of two mixed digits (20 examples shown). c) Sources computed using independent component analysis (color indicate sign). d) Sources computed using non-negative matrix factorization. e) Sources computed using linearly constrained Bayesian matrix factorization (details explained in the text).

edges. Two sources stand out: One is a black blob of approximately the same size as the digits, and another is an all white feature, which are useful for adjusting the brightness.

## 5 Conclusions

We presented a linearly constrained Bayesian matrix factorization method as well as an inference procedure for this model. On an unsupervised image separation problem, we have demonstrated that the method finds sources that have a clear an interpretable meaning. As opposed to ICA and NMF, our method finds sources that visually resemble handwritten digits.

We formulated the model in general terms, which allows specific prior information to be incorporated in the factorization. The Gaussian priors over the sources can be used if knowledge is available about the covariance structure of the sources, e.g., if the sources are known to be smooth. The constraints we used in our experiments were separate for $A$ and $B$, but the framework allows bilinearly dependent constraints to be specified, which can be used for example to specify constraints in the data domain, i.e., on the product $AB$.

As a general framework for constrained Bayesian matrix factorization, the proposed method has applications in many other areas than blind source separation. Interesting applications include blind deconvolution, music transcription, spectral unmixing, and collaborative filtering. The method can also be used in a supervised source separation setting, where the distributions over sources and mixing coefficients are learned from a training set of isolated sources. It is an interesting challenge to develop methods for learning relevant constraints from data.

# References

[1] A. T. Cemgil. Bayesian inference for nonnegative matrix factorisation models. *Computational Intelligence and Neuroscience*, 2009. doi: 10.1155/2009/785152.

[2] J. Geweke. Efficient simulation from the multivariate normal and student-t distributions subject to linear constraints and the evaluation of constraint probabilities. In *Computer Sciences and Statistics, Proceedings the 23rd Symposium on the Interface between*, pages 571–578, 1991. doi: 10.1.1.26.6892.

[3] D. D. Lee and H. S. Seung. Learning the parts of objects by non-negative matrix factorization. *Nature*, pages 788–791, October 1999. doi: 10.1038/44565.

[4] S. Moussaoui, D. Brie, A. Mohammad-Djafari, and C. Carteret. Separation of non-negative mixture of non-negative sources using a Bayesian approach and MCMC sampling. *Signal Processing, IEEE Transactions on*, 54(11):4133–4145, Nov 2006. doi: 10.1109/TSP.2006.880310.

[5] R. M. Neal. Slice sampling. *Annals of Statistics*, 31(3):705–767, 2003.

[6] G. Rodriguez-Yam, R. Davis, and L. Scharf. Efficient gibbs sampling of truncated multivariate normal with application to constrained linear regression. Technical report, Colorado State University, Fort Collins, 2004.

[7] R. Salakhutdinov and A. Mnih. Probabilistic matrix factorization. In *Neural Information Processing Systems, Advances in (NIPS)*, pages 1257–1264, 2008.

[8] R. Salakhutdinov and A. Mnih. Bayesian probabilistic matrix factorization using Markov chain Monte Carlo. In *Machine Learning, International Conference on (ICML)*, pages 880–887, 2008.

[9] M. N. Schmidt, O. Winther, and L. K. Hansen. Bayesian non-negative matrix factorization. In *Independent Component Analysis and Signal Separation, International Conference on*, volume 5441 of *Lecture Notes in Computer Science (LNCS)*, pages 540–547. Springer, 2009. doi: 10.1007/978-3-642-00599-2_68.

